# Parallelizing Model-based Reinforcement Learning Over the Sequence Length

**ZiRui Wang**
Zhejiang University, China
ziseoiwong@zju.edu.cn

**Yue Deng**
Zhejiang University, China
devindeng@zju.edu.cn

**Junfeng Long**
Shanghai AI Laboratory, China
junfengac@gmail.com

**Yin Zhang**[*]
Zhejiang University, China
zhangyin98@zju.edu.cn

## Abstract

Recently, Model-based Reinforcement Learning (MBRL) methods have demonstrated stunning sample efficiency in various RL domains. However, achieving this extraordinary sample efficiency comes with additional training costs in terms of computations, memory, and training time. To address these challenges, we propose the **Pa**rallelized **Mo**del-based **R**einforcement **L**earning (**PaMoRL**) framework. PaMoRL introduces two novel techniques: the **P**arallel **W**orld **M**odel (**PWM**) and the **P**arallelized **E**ligibility **T**race **E**stimation (**PETE**) to parallelize both model learning and policy learning stages of current MBRL methods over the sequence length. Our PaMoRL framework is hardware-efficient and stable, and it can be applied to various tasks with discrete or continuous action spaces using a single set of hyperparameters. The empirical results demonstrate that the PWM and PETE within PaMoRL significantly increase training speed without sacrificing inference efficiency. In terms of sample efficiency, PaMoRL maintains an MBRL-level sample efficiency that outperforms other no-look-ahead MBRL methods and model-free RL methods, and it even exceeds the performance of planning-based MBRL methods and methods with larger networks in certain tasks.

## 1 Introduction

Model-based Reinforcement Learning (MBRL) is widely believed to have the great potential to substantially enhance sample efficiency by training a policy through a learned world model [1, 2, 3]. Previous studies [4, 5, 3, 6] achieve the same asymptotic performance as their model-free counterparts while requiring orders of magnitude less interactions. In particular, some recent works have even achieved human-level efficiency in complex RL domains like Atari [7, 8, 9] and robot control [10, 11].

MBRL methods can be generally divided into two stages: model learning and policy learning. During the model learning stage, a parameterized world model is required to predict the environmental dynamics by constructing specific self-supervised learning tasks. The policy learning stage benefits from synthetic interactions between the policy and the world model, hence on-policy actor-critic methods or planning methods such as Model Predictive Path Integral (MPPI) [12, 13] or Monte-Carlo Tree Search (MCTS) [7, 9] can be used for policy improvement. To obtain better performance, techniques like sequential modeling and ensembling are frequently used in the model learning stage, while the policy learning stage mostly involves the computation of eligibility traces or multi-step returns [2, 14]. However, these powerful techniques often come with additional computations,

---

[*]Corresponding author: Yin Zhang.

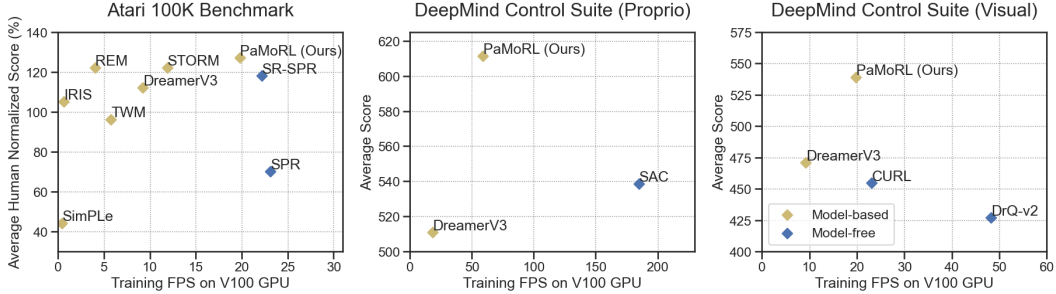

Figure 1: Comparisons on Atari 100k benchmark [16] and DeepMind Control Suite [24]. Among these methods, DreamerV3 [17], and our PaMoRL are directly evaluated on an NVIDIA V100 GPU, and IRIS [20], TWM [21], and REM [25] are evaluated on an A100 GPU, while other methods are evaluated on a P100 GPU. The extrapolation method employed aligns with the setup used in DreamerV3, where it assumes the P100 is twice as slow and the A100 is twice as fast.

memories, and training time. This leads users to carefully consider which specific MBRL method to use or even whether to use an MBRL method based on the computational resources available.

In recent years, numerous endeavors have been made to develop an efficient world model architecture. Recurrent Neural Networks (RNNs) are frequently employed as the foundational architecture for world models [15, 16, 3, 6, 17]. However, the recurrent nature of RNNs hinders parallelization, leading to slow training speeds. In contrast, transformers have emerged as a potential successor, garnering acclaim for their remarkable performance in language modeling tasks and parallelized training paradigm [18]. Several attempts have been made to incorporate transformers into world models [19, 20, 21, 22]. However, the quadratic complexity of transformers w.r.t. sequence length limits their efficiency during training and inference. To achieve an RNN-level inference efficiency, extra tricks such as half-precision training or KV-Cache are required[23]. Furthermore, none of the aforementioned works have introduced improvements in the hardware efficiency of policy learning.

In this paper, we aim to mitigate the curse of computational inefficiency of current MBRL methods and achieve the best of both worlds in terms of hardware efficiency and sample efficiency. The key idea is to fully parallelize the computations of sequential data, which has been a main workhorse of the rapid progress in deep learning over the past decade [26]. We achieve this by introducing the parallel scan. Specifically, We delve into two classic and widely implemented parallel scanners [27, 28], which can be applied for parallel training by excluding non-linear dependencies [29, 30]. Motivated by recent works in efficient sequential modeling [31, 32, 33], we observe that model architectures like linear attentions and linear RNNs not only enable parallel training but also recurrent inference. We also observe that the computations of eligibility trace estimation [2, 14] can be naturally parallelized over the sequence length by using parallel scan.

To this end, we introduce the **Pa**rallelized **Mo**del-based **R**einforcement **L**earning (**PaMoRL**) framework, which consists of two novel techniques as shown in Figure 2 that can parallelize the current MBRL paradigm over sequence length: (1) the **P**arallel **W**orld **M**odel (**PWM**) and (2) the **P**arallelized **E**ligibility **T**race **E**stimation (**PETE**). The resulting framework, PaMoRL, is hardware-efficient and stable. It is compatible with various on-policy RL methods and can be applied to both discrete and continuous control problems using a single set of hyperparameters.

We evaluated our PaMoRL framework in the Atari 100K benchmark [16] and the DeepMind Control suite [24]. Tasks in these domains include discrete and continuous action spaces, images, and proprioception observations. We choose to follow the DreamerV3 [17] paradigm, which relies on "imagination" for policy learning. The summarized experimental results are shown in Figure 1. The empirical results demonstrate that PaMoRL, despite being a framework that incorporates autoencoding, still benefits greatly from the implementation of dual parallelization techniques (i.e., PWM and PETE). These techniques substantially enhance training speed, allowing PaMoRL to rival the performance of model-free RL methods without decoders [34]. In terms of sample efficiency, PaMoRL outperforms other no-look-ahead MBRL methods and model-free RL methods. It is worth mentioning that PaMoRL even outperforms the planning-based MBRL methods or methods with much larger networks in certain tasks [8].

Our contributions can be summarized as follows:

● We introduce PaMoRL, a novel MBRL framework equipped with PWM and PETE that parallelizes both model and policy learning stages over the sequence length simultaneously.

● We evaluate our PaMoRL on the Atari 100k benchmark and DMControl suite with recent methods and obtain excellent results in terms of both sample and hardware efficiency. In addition, we conduct ablation studies on the validity of different modules, scanners, and other components.

● To the best of our knowledge, we are the first to point out that the computational process of eligibility traces can be parallelized over the sequence length. This technique can not only accelerate the value estimation process of various MBRL methods but any return-based reinforcement learning methods such as TD-$\lambda$ [2], Retrace [35] and GAE [36] can benefit from it.

## 2    Background

**Model-based Reinforcement Learning.**    We follow the paradigm of Partially Observable Markov Decision Process (POMDP) with observations $o_t$, scalar rewards $r_t$, actions $a_t$, continuation flag $c_t \in \{0, 1\}$, discount factor $\gamma \in (0, 1)$, and environmental dynamics $o_t, r_t, c_t \sim p(o_t, r_t, c_t | o_{<t}, a_{<t})$. The objective of the Reinforcement Learning (RL) is to train a policy $\pi$ that maximizes the return $\sum_{t=1}^{\infty} \gamma^{t-1} r_t$. In Model-based Reinforcement Learning (MBRL), the RL agent learns a model of the environmental dynamics through an iterative process that involves collecting data using a policy, training a model of the environment based on the accumulated data, and optimizing the policy using the learned model [1, 2, 14].

**Parallel Scan.**    As a universal parallel algorithm building block, the computations of parallel scan involve repeated application of a binary operator $\oplus$ over sequential data arrays. Previous work[37] describes scan as a good example of a computation that seems inherently sequential, but for which there is an efficient parallel algorithm. The scan of $\oplus$ with initial value $a_0$ is defined in Equation 1.

$$\text{SCAN}(\oplus, [a_1, a_2, ..., a_n], a_0) := [(a_1 \oplus a_0), (a_2 \oplus a_1 \oplus a_0), ..., (a_n \oplus a_{n-1}... \oplus a_1 \oplus a_0)] \quad (1)$$

First-order linear recurrences $h_t := (A_t \otimes h_{t-1}) \oplus x_t$ can be parallelized over the sequence length with the utilization of parallel scans if the following three conditions are met:

● $\oplus$ **is associative:** $(a \oplus b) \oplus c = a \oplus (b \oplus c)$.

● $\otimes$ **is semi-associative:** there exists a binary associative operator $\odot$ such that $a \otimes (b \otimes c) = (a \odot b) \otimes c$.

● $\otimes$ **distributes over** $\oplus$**:** $a \otimes (b \oplus c) = (a \otimes b) \oplus (a \otimes c)$.

We observe vector addition $a \oplus b := a + b$, matrix-vector multiplication $A \otimes b := A \cdot b$, and matrix-matrix multiplication $A \odot B := A \cdot B$ fulfill the aforementioned conditions. This allows the parallel computation of $x_t := (A_t \cdot x_{t-1}) + b_t$ across time steps $t$, considering input vectors $b_t$ and square matrices $A_t$. Considering the operators required in computing linear attentions [31] and eligibility trace estimations [2, 14] involve only diagonal matrices, the linear recurrence can be re-formulated as $x_t := \lambda_t \odot x_{t-1} + b_t$, where $\lambda_t$ is the eigenvalues of the diagonal matrices and $\odot$ is an element-wise multiplication.

## 3    Methodology

We introduce our **Pa**rallelized **Mo**del-based **R**einforcement **L**earning (**PaMoRL**) framework, which facilitates dual parallelization across both model and policy learning stages. By parallelized training and recurrent inference, PaMoRL significantly improves training speed while avoiding additional computation overhead during inference. Figure 2 illustrates the overview of our PaMoRL framework, and we will now proceed to elaborate on its details.

### 3.1    Parallelized World Model Learning.

**World Model Architecture Overview.**    As with other MBRL methods, our world model is trained to predict environmental dynamics. Since observations can be high-dimensional (e.g., images), we

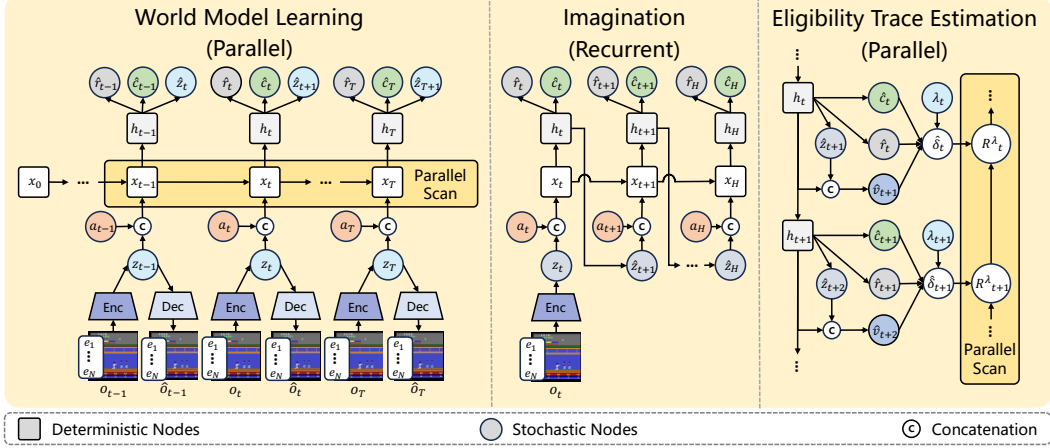

Figure 2: Overview of our PaMoRL framework. The symbols used in the figure are explained in Sections 3.1 and Section 3.2. The computations of the sequential model's outputs and the TD-$\lambda$ returns allow using parallel scans. In contrast, the imaginations cannot be parallelized over the sequence length because a non-linear actor network is required for action sampling.

predict future representations rather than future observations. This reduces accumulating errors and enables massively parallel training with a large batch size. The compact representations are obtained by an autoencoder and can be utilized to predict future observations, reward, and continuation flags.

To exclude non-linear dependencies for parallel training and obtain better performance, we make several modifications to the vanilla Recurrent State-Space Model's (RSSM) [3, 6, 17] configurations: (1) differentiating the hidden states $x_t$ from the sequential model's outputs $h_t$, (2) excluding $h_t$ from the inputs of the encoder and decoder, (3) eliminating the stochastic states $z_t$ from the predictors' inputs, and (4) applying Batch Normalization for the encoder and dynamic predictor's outputs before the distributions are computed. Similar to RSSM, our model consists of six components:

$$
\begin{array}{ll}
\text{Encoder: } z_t \sim q_\theta(z_t|o_t) & \text{Decoder: } \hat{o}_t \sim p_\theta(\hat{o}_t|z_t) \\
\text{Sequence model: } h_t, x_t = f_\theta(x_{t-1}, z_{t-1}, a_{t-1}) & \text{Dynamics predictor: } \hat{z}_t \sim p_\theta(\hat{z}_t|h_t) \\
\text{Reward predictor: } \hat{r}_t \sim p_\theta(\hat{r}_t|h_t) & \text{Continue predictor: } \hat{c}_t \sim p_\theta(\hat{c}_t|h_t)
\end{array}
\tag{2}
$$

The encoder and decoder use convolutional neural networks (CNN) for image inputs and multi-layer perceptrons (MLPs) for proprioception inputs. The sequence model has multiple stacked residual blocks, each of which consists of a modified linear attention [31] module and a Gated Linear Unit (GLU) [38] module. The dynamics, reward, and continue predictors are all MLPs. Consistent with previous work [6, 17], we set the $q_\theta(z_t|o_t)$ as a stochastic distribution comprising 32 categories, each with 32 classes, and we take straight-through gradients through the sampling step [39].

**Sequence Model Architectures.** As mentioned above, each residual block of our sequence model consists of a modified linear attention module and a GLU module. The vanilla linear attention module, as introduced in previous work [31], employs $1 + \text{ELU}$ as an element-wise kernel function applied to queries $q_t$ and keys $k_t$ taking $u_t$ as input. This configuration allows for its reformulation into an RNN-style recurrent form. However, this version of linear attention is prone to unstable convergence during training due to the unbounded gradients [40]. Thus, we remove the time-dependent normalizer, which is designed to approximate the Softmax operator and use an RMSNorm [41] for stabilize training. Furthermore, we incorporate the token mixing module from RWKV [32], which accepts inputs $u_t$ and previous inputs $u_{t-1}$, along with the gating mechanism in Gated Recurrent Unit (GRU) [42] to provide an input-dependent decay rate $g_t$ for hidden state $x_t$. The subsequent GLU module selects SiLU as the gating function, taking linear attention output $y_t$ as input. By integrating all the modifications, we can derive the entire block of the sequence model as shown in Equation 3.

$$q_t, k_t = 1 + \text{ELU}(u_t W_q), 1 + \text{ELU}(u_t W_k),$$
$$v_t = \text{Sigmoid}(u_t W_r) \odot u_t W_v,$$
$$g_t = \text{Sigmoid}((\mu \odot u_t + (1 - \mu) \odot u_{t-1})W_g),$$
$$x_t = g_t \odot x_{t-1} + k_t^\top v_t, \tag{3}$$
$$y_t = \text{RMSNorm}(q_t x_t)W_h + u_t,$$
$$h_t = \text{SiLU}(y_t W_g) \odot y_t W_y + y_t.$$

The architecture of our modified linear attention satisfies the conditions in Section 2 and can be effectively computed using parallel scans. We can refer to Table 1 to summarize the computational complexities of various model architectures such as vanilla attention, RNN, SSM, and our modified linear attention in the training, inference, and imagination stages.

**Loss Functions.** The total loss function of model learning is shown as in Equation 4, where $\beta_{\text{pred}}$, $\beta_{\text{rep}}$, and $\beta_{\text{dyn}}$ are coefficients to adjust the influence of each term in the loss function [43, 17].

$$\mathcal{L}(\theta) \quad = \mathbb{E}_{q_\theta} \Big[ \sum_{t=1}^{T} \beta_{\text{pred}} \mathcal{L}^{\text{pred}}(\theta, h_t, o_t, r_t, c_t, z_t) + \beta_{\text{rep}} \mathcal{L}^{\text{rep}}(\theta, h_t, o_t) + \beta_{\text{dyn}} \mathcal{L}^{\text{dyn}}(\theta, h_t, o_t) \Big]$$
$$\mathcal{L}^{\text{pred}}(\theta) \quad = -\ln p_\theta(r_t|h_t) - \ln p_\theta(c_t|h_t) + ||\hat{o}_t - o_t||_2 \tag{4}$$
$$\mathcal{L}^{\text{rep}}(\theta) \quad = \max(1, \text{KL}[q_\theta(z_t|o_t) \ || \ \text{sg}(p_\theta(\hat{z}_t|h_t))])$$
$$\mathcal{L}^{\text{dyn}}(\theta) \quad = \max(1, \text{KL}[\text{sg}(q_\theta(z_t|o_t)) \ || \ p_\theta(\hat{z}_t|h_t)])$$

The operation $\text{sg}(\cdot)$ represents the stop gradient operation. The KL divergences are derived from the Evidence Lower Bound (ELBO). We clip the KL divergence when it falls below the threshold of 1 [6, 17] and use the KL-balancing trick to prioritize the training losses [17].

## 3.2 Policy Learning

The policy learning stage incorporates the actor and critic networks, both of which are MLPs, taking concatenation of $z_t$ and $h_t$ as input state $s_t$.

$$\text{Actor: } a_t \sim \pi_\phi(a_t|s_t), \quad \text{Critic: } v_\psi(s_t). \tag{5}$$

Our policy learning method is in line with DreamerV3 [17] and can be used for both discrete and continuous action spaces. The critic uses TD-$\lambda$ [2] as the its target, as shown in Equation 6, where $\hat{r}_t$ represents the reward predicted by the world model, and $\hat{c}_t$ represents the predicted continuation flag.

$$R_t^\lambda = \hat{r}_t + (\gamma \hat{c}_t) \left[ (1 - \lambda)v_\phi(s_{t+1}) + \lambda R_{t+1}^\lambda \right]$$
$$= (\lambda \gamma \hat{c}_t)R_{t+1}^\lambda + \left[ \hat{r}_t + (1 - \lambda)(\gamma \hat{c}_t)v_\phi(s_{t+1}) \right], \quad R_T^\lambda = v_T \tag{6}$$

The actor utilizes the Reinforce estimator [44] to compute the actor loss with a fixed entropy regularization term. The complete loss is described by Equation 7.

$$\mathcal{L}(\phi) = -\sum_{t=1}^{T} \text{sg}\left(\frac{R_t^\lambda - v_\psi(s_t)}{\max(1, S)}\right) \log \pi_\phi(a_t|s_t) - \eta H(\pi_\phi(a_t|s_t))$$
$$\mathcal{L}(\psi) = -\sum_{t=1}^{T} (v_\psi(s_t) - \text{sg}(R_t^\lambda))^2. \tag{7}$$

The hyper-parameter $\eta$ represents the coefficient of the entropy regularization term. The normalization ratio $S$ utilized in the actor loss is defined in Equation 8, which is computed as the range between the 95th and 5th percentiles of the TD-$\lambda$ returns $R_t^\lambda$ across the batch.

Table 1: The step complexities [28] of different architectures, where $L$ is the sequence length and $H$ is the imagination horizon. Attention considers the full context with a burn-in and imagined steps of $\mathcal{O}(L+H)$, leading to a complexity of $\mathcal{O}((L+H)^2)$. It is worth noting that the SSMs in recent works [46, 47] do not incorporate any gating mechanism or selectivities. Thus, despite SSMs and linear attentions both achieving the minimum complexity, linear attentions remain more expressive.

| Architecture | Training | Inference step | Imagination step | Parallel | Resettable | Selective |
|---|---|---|---|---|---|---|
| Atten | $\mathcal{O}(L^2)$ | $\mathcal{O}(L^2)$ | $\mathcal{O}((L+H)^2)$ | ✓ | ✓ | ✓ |
| RNN | $\mathcal{O}(L)$ | $\mathcal{O}(1)$ | $\mathcal{O}(1)$ | ✗ | ✓ | ✓ |
| SSM (FFT) | $\mathcal{O}(L\log L)$ | $\mathcal{O}(1)$ | $\mathcal{O}(1)$ | ✓ | ✗ | ✗ |
| SSM (Scan) | $\mathcal{O}(L)$ | $\mathcal{O}(1)$ | $\mathcal{O}(1)$ | ✓ | ✓ | ✗ |
| Lin-Atten (Scan) | $\mathcal{O}(L)$ | $\mathcal{O}(1)$ | $\mathcal{O}(1)$ | ✓ | ✓ | ✓ |

$$S = \text{percentile}(R_t^\lambda, 95) - \text{percentile}(R_t^\lambda, 5) \tag{8}$$

By rearranging Equation 6, we can see that the calculations of both TD-$\lambda$ and Retrace returns also also meet the conditions mentioned in Section 2. Therefore, they can be efficiently computed using parallel scan. This observation also applies to other eligibility trace estimation methods such as GAE [36] and Retrace [35], as they still satisfy the aforementioned conditions.

### 3.3 Parallel Scan Algorithms

In both the model learning stage and the policy learning stage, we use two different parallel scanners: the Kogge-stone scanner [45] and the Odd-even scanner [28].

The Kogge-stone scanner [45] is commonly used in hardware design for adders. It has a computational complexity of $\mathcal{O}(L\log_2 L)$ for sequence length $L$ and a step complexity of $\mathcal{O}(\log_2 L)$ after full parallelization. This indicates that it has higher computational redundancy, lower running time, and sufficient computational resources, making it suitable for parallel computation in a small batch.

The Odd-even scanner [28] is based on the concept of binary balanced trees. It has a computational complexity of $\mathcal{O}(2L)$ for a sequence length $L$ and a step complexity of $\mathcal{O}(2\log_2 L)$ after being fully parallelized. Despite theoretically taking more steps than the Kogge-stone scanner, it offers lower computational complexity and more uniform load sharing, making it better suited for large-scale parallel computation. Further details and illustrations are in Appendix B.

## 4 Experiments

In this section, we aim to evaluate both the sample and training efficiency of our PaMoRL framework on the Atari 100K benchmark [16] and the DMControl suite [24]. The tasks include various scenarios with image and proprioception observations and discrete and continuous action spaces.

### 4.1 Experimental Setup

**Atari 100K.** Atari 100K consists of 26 video games with discrete action dimensions of up to 18. The 100K samples are equated to 400K actual game frames, corresponding to approximately 2 hours of real-time gameplay, with action repeats of 4. The human normalized score is defined as $(\text{score}_{\text{agent}} - \text{score}_{\text{random}})/(\text{score}_{\text{human}} - \text{score}_{\text{random}})$, where $\text{score}_{\text{random}}$ comes from a random policy, and $\text{score}_{\text{human}}$ is obtained from human players [48].

**DeepMind Control Suite.** DeepMind Control Suite consists of various control tasks with continuous action spaces. Referring to the categorizations in Sample MuZero [49] and EfficientZero V2 [9], tasks are divided into **easy** and **hard** categories. We followed the experimental setup of EfficientZero V2 [9] and established two benchmarks, named **Proprio Control** and **Visual Control**.

Among them, **Proprio Control** uses proprioception observations with 50K training samples for easy tasks and 100K for hard tasks, and **Visual Control** uses image observations with 100K training

Table 2: Experimental results on the 26 games of Atari 100k after 2 hours of real-time experience and human-normalized aggregate metrics. Bold and underlined numbers indicate the highest and the second-highest scores, respectively. PaMoRL outperforms other methods regarding the number of superhuman games, mean, and median.

| Game | Random | Human | SPR | SR-SPR | SimPLe | IRIS | TWM | STORM | DreamerV3 | PaMoRL (Ours) |
|---|---|---|---|---|---|---|---|---|---|---|
| Alien | 227.8 | 7127.7 | 801.5 | 1015.5 | 616.9 | 420 | 674.6 | 984 | 959 | **1270.6** |
| Amidar | 5.8 | 1719.5 | 176.3 | 203.1 | 88 | 143 | 121.8 | 205 | 139 | **264.4** |
| Assault | 222.4 | 742 | 571 | 1069.5 | 527.2 | **1524.4** | 682.6 | 801 | 706 | 883.8 |
| Asterix | 210 | 8503.3 | 977.8 | 916.5 | 1128.3 | 853.6 | 1116.6 | 1028 | 932 | **2957.3** |
| BankHeist | 14.2 | 753.1 | 380.9 | 472.3 | 34.2 | 53.1 | 466.7 | 641 | 649 | 255.9 |
| BattleZone | 2360 | 37187.5 | 16651 | 19398.4 | 5184.4 | 13074 | 5068 | 13540 | 12250 | **23120** |
| Boxing | 0.1 | 12.1 | 35.8 | 46.7 | 9.1 | 70.1 | 77.5 | 80 | 78 | **87.9** |
| Breakout | 1.7 | 30.5 | 17.1 | 28.8 | 16.4 | **83.7** | 20 | 16 | 31 | 15.8 |
| ChopperCommand | 811 | 7387.8 | 974.8 | **2201** | 1246.4 | 1565 | 1697.4 | 1888 | 420 | 2110.7 |
| CrazyClimber | 10780.5 | 35829.4 | 42923.6 | 43122.3 | 62583.6 | 59324.2 | 71820.4 | 66776 | **97190** | 84102 |
| DemonAttack | 152.1 | 1971 | 545.2 | **2898.1** | 208.1 | 2034.4 | 350.2 | 165 | 303 | 208.2 |
| Freeway | 0 | 29.6 | 24.4 | 24.9 | 20.3 | 31.1 | 24.3 | 0 | 0 | **33.8** |
| Frostbite | 65.2 | 4334.7 | 1821.5 | 1752.8 | 254.7 | 259.1 | 1475.6 | 1316 | 909 | **3711.4** |
| Gopher | 257.6 | 2412.5 | 715.2 | 711.2 | 771 | 2236.1 | 1674.8 | **8240** | 3730 | 5085.2 |
| Hero | 1027 | 30826.4 | 7019.2 | 7679.6 | 2656.6 | 7037.4 | 7254 | 11044 | 11161 | **12076.2** |
| Jamebond | 29 | 302.8 | 365.4 | 392.8 | 125.3 | 462.7 | 362.4 | **509** | 445 | 405 |
| Kangaroo | 52 | 3035 | 3276.4 | 3254.9 | 323.1 | 838.2 | 1240 | **4208** | 4098 | 2554.7 |
| Krull | 1598 | 2665.5 | 3688.9 | 5824.8 | 4539.9 | 6616.4 | 6349.2 | **8413** | 7782 | 7273.2 |
| KungFuMaster | 258.5 | 22736.3 | 13192.7 | 17095.6 | 17257.2 | 21759.8 | 24554.6 | **26182** | 21420 | 24624.7 |
| MsPacman | 307.3 | 6951.6 | 1313.2 | 1522.6 | 1480 | 999.1 | 1588.4 | **2673** | 1327 | 2201.7 |
| Pong | -20.7 | 14.6 | -5.9 | -3 | 12.8 | 14.6 | **18.8** | 11 | 18 | 15.5 |
| PrivateEye | 24.9 | 69571.3 | 124 | 95.8 | 58.3 | 100 | 86.6 | **7781** | 882 | 4968.6 |
| Qbert | 163.9 | 13455 | 669.1 | 3850.6 | 1288.8 | 745.7 | 3330.8 | **4522** | 3405 | 4730.3 |
| Roadrunner | 11.5 | 7845 | 14220.5 | 13623.5 | 5640.6 | 9614.6 | 9109 | 17564 | 15565 | **24726.7** |
| Seaquest | 68.4 | 42054.7 | 583.1 | **800.5** | 683.3 | 661.3 | 774.4 | 525 | 618 | 595.2 |
| UpNDwon | 533.4 | 11693.2 | 28138.5 | **95501.1** | 3350.3 | 3546.2 | 15981.7 | 7985 | 7667 | 11935.8 |
| Games >Human | 0 | 26 | 7 | 9 | 2 | 9 | 8 | 9 | 9 | **11** |
| Median | 0% | 100% | 41.53% | 56.07% | 14% | 29% | 51% | 42.63% | 49% | **71.75%** |
| Mean | 0% | 100% | 70.34% | 118.84% | 44% | 105% | 96% | 122.30% | 112% | **126.64%** |

samples for easy tasks and 200K for hard tasks. Each benchmark includes 16 tasks. Action repeats are set to 2, and the maximum episode length is 1000 for both benchmarks, in line with previous studies [17, 13, 9]. We choose various baselines for each domain, which include SAC [50], DrQ-v2 [51], and DreamerV3 [17].

## 4.2 Experimental Results

In this section, we do not compare our results with look-ahead search methods [52, 7, 12, 13] or methods using larger networks [8], as our main goal in terms of sample efficiency is to improve performance while **maximizing the hardware efficiency** of existing MBRL methods.

**Atari 100K.** The summarized results are shown in Figure 4. The full results for individual games in the Atari 100k benchmark are elaborated in Table 2, where scores are normalized against those of human players. Our PaMoRL framework attains a mean score of **126.64%** and a median score of **71.75%**, surpassing the other methods in terms of both mean and median human normalized score. For detailed training curves, please refer to Appendix C. Additionally, you can find more results and further discussions, including methods with look-ahead search or larger networks, in Appendix I.

**DeepMind Control Suite.** Table 3 shows that our method achieves a mean score of **661.2** across 16 tasks. As shown in Table 3, our method achieves a mean score of **661.2** using proprioception observations and **538.7** using image observations across 16 tasks, surpassing the previous state-of-the-art, DreamerV3. The improvement in sample efficiency is attributed to two key modules: the token mixing module in the PWM, where the extra previous input provides more information to the data-dependent decay rate, and the implementation of RMSNorm, which improves the stability of the learning of the linear attention module, especially in the case of limited data. Our PaMoRL framework consistently demonstrates MBRL-level sample efficiency in tasks with proprioception observations, image observations, and discrete and continuous action spaces. Detailed training curves can be found in Figure 9 and Figure 10 in Appendix D.

Table 3: Experimental results on the DeepMind Control suite. Bold and underlined numbers indicate the highest and the second-highest scores, respectively. PaMoRL outperforms other baselines in terms of the number of mean and median scores.

| Task | Proprio Control | | | Vision Control | | | |
|---|---|---|---|---|---|---|---|
| | SAC | DreamerV3 | PaMoRL (Ours) | CURL | DrQ-v2 | DreamerV3 | PaMoRL (Ours) |
| Cartpole Balance | **997.6** | 839.6 | 994.7 | 963.3 | **965.5** | 956.4 | 610.3 |
| Cartpole Balance Sparse | 993.1 | 559 | **997.4** | 999.4 | **1000** | 813 | 996.5 |
| Cartpole Swingup | **861.6** | 527.7 | 773.6 | **765.4** | 756 | 374.8 | 281.9 |
| Cup Catch | 949.9 | 729.6 | **957.9** | 932.3 | 468 | 947.7 | **966.3** |
| Finger Spin | **900** | 765.8 | 835.8 | **850.2** | 459.4 | 633.2 | 765.3 |
| Pendulum Swingup | 158.9 | **830.4** | 707.1 | 144.1 | 233.3 | **619.3** | 26.6 |
| Reacher Easy | 744 | 693.4 | **761.6** | 467.9 | 722.1 | 441.4 | **950.2** |
| Reacher Hard | 646.5 | **768** | 645.9 | 112.7 | **202.9** | 120.4 | 103.7 |
| Cartpole Swingup Sparse | 256.6 | 172.7 | **542.3** | 8.8 | 81.2 | **392.4** | 263.6 |
| Cheetah Run | **680.9** | 400.8 | 313.2 | 405.1 | 418.4 | 587.3 | **935.6** |
| Finger Turn Easy | **630.8** | 560.5 | 617.1 | 371.5 | 286.8 | 366.6 | **886.2** |
| Finger Turn Hard | 414 | **474.2** | 389.7 | 236.3 | 268.4 | 258.5 | **500.1** |
| Hopper Hop | 0.1 | 9.7 | **387.5** | 84.5 | 26.3 | 76.3 | **426.9** |
| Hopper Stand | 3.8 | **296.1** | 151.5 | 627.7 | 290.2 | **652.5** | 189.7 |
| Quadruped Run | 139.7 | **289** | 246.7 | 170.9 | 339.4 | 168 | **344.8** |
| Quadruped Walk | 237.5 | 256.2 | **457.9** | 131.8 | 311.6 | 122.6 | **371.6** |
| Mean | 538.4 | 510.8 | **611.2** | 454.5 | 426.8 | 470.7 | **538.7** |
| Median | **638.7** | 543.4 | 631.5 | 388.3 | 325.5 | 416.9 | **463.5** |

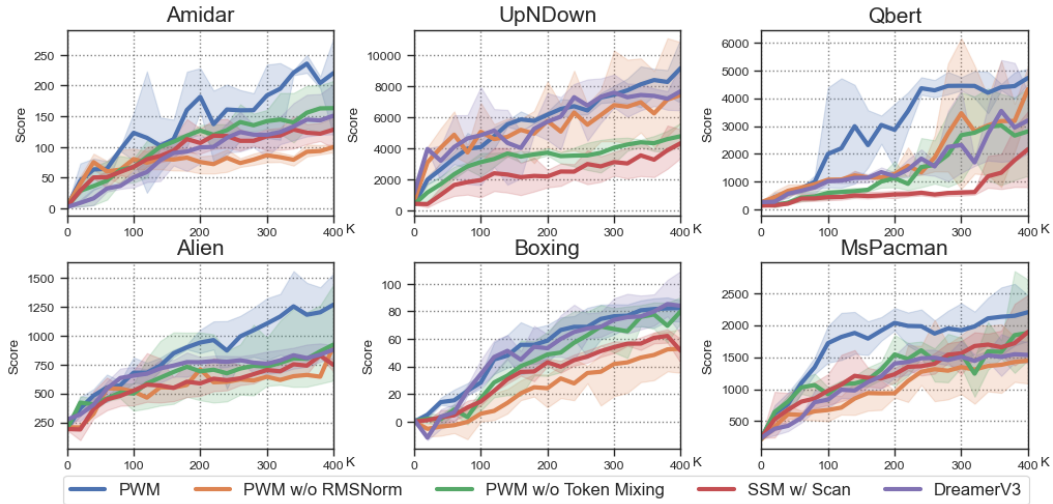

Figure 3: Ablation studies of the effectiveness of each module of PWM, where SSM is equivalent to removing the data-dependent decay rate of PWM. We also include vanilla DreamerV3 as a baseline.

## 4.3 Ablation Study

In this section, we will be conducting ablation studies to evaluate the effectiveness of PWM and PETE in terms of stabilizing training and improving hardware efficiency. For more details, including PyTorch-style pseudo-code, please refer to Appendix G.

**World Model Design.** The results presented in Figure 3 demonstrate the impact of adding or removing the token mixing, RMSNorm, and data-dependent decay rate in various games in the Atari 100K benchmark. To showcase the benefits of token mixing in sequence prediction, we focused on tasks such as *Alien*, *Boxing*, and *MsPacman*. Additionally, we measured the improvement of RMSNorm on training stability by considering tasks like *Amidar*, *UpNDown*, and *Qbert*.

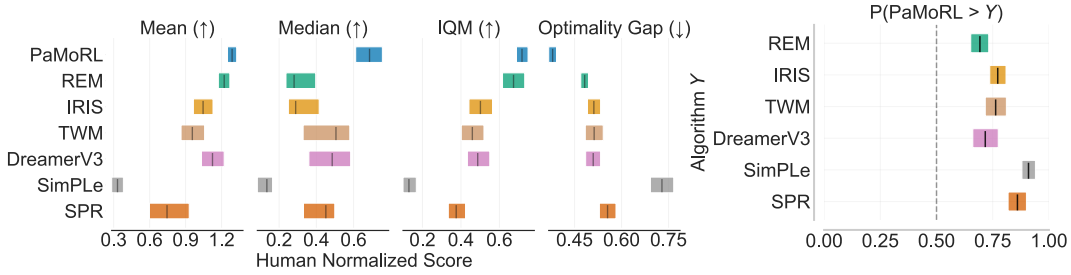

Figure 4: **(Left)** Atari 100K aggregated metrics with 95% stratified bootstrap confidence intervals of the mean, median, and interquartile mean (IQM) human-normalized scores and optimality gap. **(Right)** Probabilities of improvement, i.e. how likely it is for our PaMoRL to outperform baselines.

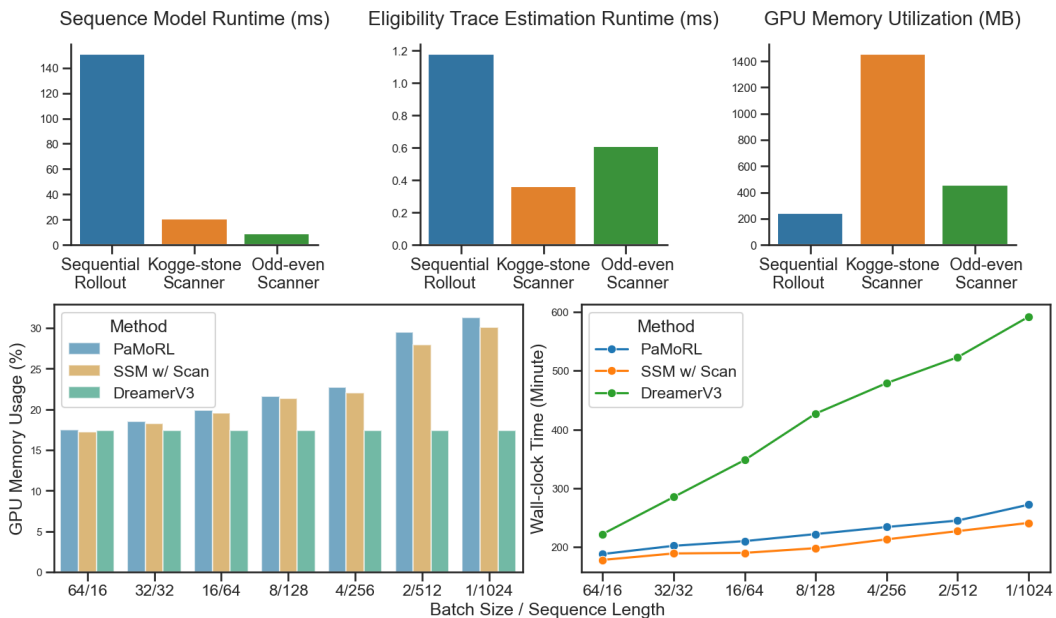

Figure 5: **(Upper)** Comparison of parallel scanners with sequential rollout in terms of runtime for sequence modeling and eligibility trace estimation, as well as total GPU memory utilization. **(Lower)** Wall-clock time vs. GPU memory usage comparison for our PaMoRL method, SSM, and DreamerV3 across various batch size and sequence length combinations.

The findings in Figure 3 indicate that, while the token mixing module has minimal impact on the final performance for tasks where the reward can be accurately predicted from a single frame (e.g., *Boxing*), it leads to a performance drop on tasks that require several contextual frames to predict the reward accurately (e.g., *Alien* and *Ms. Pacman*). Regarding RMSNorm, removing it negatively affects the final performance and increases the instability of the training process.

There are two possible reasons for this difference. First, the gradient is bounded after the original normalizer is removed [40]. Adding RMSNorm further enhances training stability, which is especially important in the setting of limited data and end-to-end training. Second, RMSNorm only rescales the input and maintains the original center of the samples, which allows the module's output to maximize the information's retention.

**Parallel Scanner Selection.** Figure 5 shows PWM and PETE's runtime and GPU memory utilization on a single 3090 GPU using different scanners, respectively. Sequence model computation

achieves $7.2\times$ and $16.6\times$ speedups compared to sequential rollout using the Kogge-stone and Odd-even scanners, respectively, with a sequence length of 64. In this case, the Kogge-stone scanner with the theoretically lowest runtime takes more than the Odd-even scanner in practice. This is because the computation of the sequence model involves the parallelism of both batch and hidden dimensions, which belongs to massively parallel computation, and the Kogge-stone scanner cannot realize full parallelism and thus encounters a bottleneck in computational resources. In contrast, the Odd-even scanner is due to less computational redundancy, which allows the computational process of sequence modeling to achieve full parallelism and thus spends less running time. The PETE uses the Kogge-stone scanner and Odd-even scanner to achieve $3\times$ and $2\times$ speedups, respectively, with a sequence length of 16. Since the eligibility trace has a dimension of only 1, the Kogge-stone scanner can take full advantage of it. It thus achieves less runtime compared to the Odd-even scanner.

Regarding GPU memory utilization, using the Kogge-stone scanner imposes an additional $6\times$ overhead compared to the sequential rollout, while the Odd-even scanner imposes an additional $2\times$ overhead compared to the sequential rollout. However, the additional GPU memory overhead of parallel computation is not significant compared to the GPU memory overhead of encoder and decoder computation, especially in tasks with image observation.

Therefore, we recommend using the Odd-even scanner for PWM and the Kogge-stone scanner for PETE to achieve maximal speed with acceptable additional GPU memory utilization.

**Batch Normalization Trick.** World models are commonly learned using variational autoencoders to create concise representations of observations. However, they have some drawbacks, such as the tendency to disregard small moving objects. In Figure 11 in Appendix K, the reconstruction results are compared with and without using Batch Normalization for the *Pong* and *Breakout* games in the Atari 100K benchmark. It is observed that Batch Normalization improves the ability to distinguish similar video frames and capture information about small objects by re-centering the samples.

Additionally, Figure 12 demonstrates that PWM benefits from the batch normalization trick, whereas DreamerV3 does not. This is likely due to PWM's decoder solely having stochastic states as inputs, making it challenging for training samples to be distinguished from each other in the early stages of training, leading to "posterior collapse" [53]. On the other hand, DreamerV3's decoder mitigates this problem by incorporating additional deterministic states as conditional inputs.

## 5    Conclusion & Limitations

In this paper, we introduce the PaMoRL framework, an MBRL method capable of being computed using the parallel scan in both the model learning and policy learning stages. The key breakthrough of PaMoRL is the integration of two novel techniques: the Parallelized World Model and Parallelizable Eligibility Trace Estimation. With these techniques, we simultaneously accelerate the training process while maintaining MBRL-level sample efficiency. PaMoRL demonstrates excellent hardware efficiency and training stability in various games or tasks in the Atari 100K benchmark and DeepMind Control suite without incurring additional overhead during inference. An important contribution of our work is the introduction of a modified linear attention module in the MBRL method. Furthermore, we show that eligibility trace estimation computation can be parallelized for the first time.

It's important to acknowledge the limitations of our work. For instance, planning-based MBRL methods cannot parallelize computation over the sequence length, which hinders the incorporation of the most sample-efficient methods within our PaMoRL framework to maximize hardware efficiency. It would be interesting to explore using hybrid architectures to enhance PaMoRL by leveraging the strengths of Transformers, RNNs, and SSMs. Additionally, the world model and baselines used for comparison in PaMoRL are trained end-to-end with joint optimization of the image encoder and sequence model. While this end-to-end training paradigm enables the world model to predict the latent representations, it also impacts the scalability of the world model.

## Acknowledgments and Disclosure of Funding

This work was supported by the NSFC project (No. 62072399), Zhejiang Provincial Natural Science Foundation of China under Grant No. LZ23F020009, Chinese Knowledge Center for Engineering Sciences and Technology, MoE Engineering Research Center of Digital Library, China Research Centre on Data and Knowledge for Engineering Sciences and Technology, and the Fundamental Research Funds for the Central Universities (No. 226-2024-00170).

## References

[1] Richard S Sutton. Dyna, an integrated architecture for learning, planning, and reacting. *ACM Sigart Bulletin*, 2(4):160–163, 1991.

[2] Richard S Sutton and Andrew G Barto. Reinforcement learning: An introduction. *Robotica*, 17(2):229–235, 1999.

[3] Danijar Hafner, Timothy Lillicrap, Jimmy Ba, and Mohammad Norouzi. Dream to control: Learning behaviors by latent imagination. *arXiv preprint arXiv:1912.01603*, 2019.

[4] Kurtland Chua, Roberto Calandra, Rowan McAllister, and Sergey Levine. Deep reinforcement learning in a handful of trials using probabilistic dynamics models. *Advances in neural information processing systems*, 31, 2018.

[5] Michael Janner, Justin Fu, Marvin Zhang, and Sergey Levine. When to trust your model: Model-based policy optimization. *Advances in neural information processing systems*, 32, 2019.

[6] Danijar Hafner, Timothy Lillicrap, Mohammad Norouzi, and Jimmy Ba. Mastering atari with discrete world models. *arXiv preprint arXiv:2010.02193*, 2020.

[7] Weirui Ye, Shaohuai Liu, Thanard Kurutach, Pieter Abbeel, and Yang Gao. Mastering atari games with limited data. *Advances in Neural Information Processing Systems*, 34:25476–25488, 2021.

[8] Max Schwarzer, Johan Samir Obando Ceron, Aaron Courville, Marc G Bellemare, Rishabh Agarwal, and Pablo Samuel Castro. Bigger, better, faster: Human-level atari with human-level efficiency. In *International Conference on Machine Learning*, pages 30365–30380. PMLR, 2023.

[9] Shengjie Wang, Shaohuai Liu, Weirui Ye, Jiacheng You, and Yang Gao. Efficientzero v2: Mastering discrete and continuous control with limited data. *arXiv preprint arXiv:2403.00564*, 2024.

[10] Philipp Wu, Alejandro Escontrela, Danijar Hafner, Pieter Abbeel, and Ken Goldberg. Daydreamer: World models for physical robot learning. In *Conference on Robot Learning*, pages 2226–2240. PMLR, 2023.

[11] Russell Mendonca, Shikhar Bahl, and Deepak Pathak. Structured world models from human videos. *arXiv preprint arXiv:2308.10901*, 2023.

[12] Nicklas Hansen, Xiaolong Wang, and Hao Su. Temporal difference learning for model predictive control. *arXiv preprint arXiv:2203.04955*, 2022.

[13] Nicklas Hansen, Hao Su, and Xiaolong Wang. Td-mpc2: Scalable, robust world models for continuous control. *arXiv preprint arXiv:2310.16828*, 2023.

[14] Richard S Sutton and Andrew G Barto. *Reinforcement learning: An introduction*. MIT press, 2018.

[15] David Ha and Jürgen Schmidhuber. World models. *arXiv preprint arXiv:1803.10122*, 2018.

[16] Lukasz Kaiser, Mohammad Babaeizadeh, Piotr Milos, Blazej Osinski, Roy H Campbell, Konrad Czechowski, Dumitru Erhan, Chelsea Finn, Piotr Kozakowski, Sergey Levine, et al. Model-based reinforcement learning for atari. *arXiv preprint arXiv:1903.00374*, 2019.

[17] Danijar Hafner, Jurgis Pasukonis, Jimmy Ba, and Timothy Lillicrap. Mastering diverse domains through world models. *arXiv preprint arXiv:2301.04104*, 2023.

[18] Ashish Vaswani, Noam Shazeer, Niki Parmar, Jakob Uszkoreit, Llion Jones, Aidan N Gomez, Łukasz Kaiser, and Illia Polosukhin. Attention is all you need. *Advances in neural information processing systems*, 30, 2017.

[19] Chang Chen, Yi-Fu Wu, Jaesik Yoon, and Sungjin Ahn. Transdreamer: Reinforcement learning with transformer world models. *arXiv preprint arXiv:2202.09481*, 2022.

[20] Vincent Micheli, Eloi Alonso, and François Fleuret. Transformers are sample efficient world models. *arXiv preprint arXiv:2209.00588*, 2022.

[21] Jan Robine, Marc Höftmann, Tobias Uelwer, and Stefan Harmeling. Transformer-based world models are happy with 100k interactions. *arXiv preprint arXiv:2303.07109*, 2023.

[22] Weipu Zhang, Gang Wang, Jian Sun, Yetian Yuan, and Gao Huang. Storm: Efficient stochastic transformer based world models for reinforcement learning. *arXiv preprint arXiv:2310.09615*, 2023.

[23] Reiner Pope, Sholto Douglas, Aakanksha Chowdhery, Jacob Devlin, James Bradbury, Jonathan Heek, Kefan Xiao, Shivani Agrawal, and Jeff Dean. Efficiently scaling transformer inference. *Proceedings of Machine Learning and Systems*, 5, 2023.

[24] Yuval Tassa, Yotam Doron, Alistair Muldal, Tom Erez, Yazhe Li, Diego de Las Casas, David Budden, Abbas Abdolmaleki, Josh Merel, Andrew Lefrancq, et al. Deepmind control suite. *arXiv preprint arXiv:1801.00690*, 2018.

[25] Lior Cohen, Kaixin Wang, Bingyi Kang, and Shie Mannor. Improving token-based world models with parallel observation prediction. *arXiv preprint arXiv:2402.05643*, 2024.

[26] Yi Heng Lim, Qi Zhu, Joshua Selfridge, and Muhammad Firmansyah Kasim. Parallelizing non-linear sequential models over the sequence length. *arXiv preprint arXiv:2309.12252*, 2023.

[27] Richard E Ladner and Michael J Fischer. Parallel prefix computation. *Journal of the ACM (JACM)*, 27(4):831–838, 1980.

[28] Mark Harris, Shubhabrata Sengupta, and John D Owens. Parallel prefix sum (scan) with cuda. *GPU gems*, 3(39):851–876, 2007.

[29] Eric Martin and Chris Cundy. Parallelizing linear recurrent neural nets over sequence length. *arXiv preprint arXiv:1709.04057*, 2017.

[30] Jimmy TH Smith, Andrew Warrington, and Scott W Linderman. Simplified state space layers for sequence modeling. *arXiv preprint arXiv:2208.04933*, 2022.

[31] Angelos Katharopoulos, Apoorv Vyas, Nikolaos Pappas, and François Fleuret. Transformers are rnns: Fast autoregressive transformers with linear attention. In *International conference on machine learning*, pages 5156–5165. PMLR, 2020.

[32] Bo Peng, Eric Alcaide, Quentin Anthony, Alon Albalak, Samuel Arcadinho, Huanqi Cao, Xin Cheng, Michael Chung, Matteo Grella, Kranthi Kiran GV, et al. Rwkv: Reinventing rnns for the transformer era. *arXiv preprint arXiv:2305.13048*, 2023.

[33] Albert Gu and Tri Dao. Mamba: Linear-time sequence modeling with selective state spaces. *arXiv preprint arXiv:2312.00752*, 2023.

[34] Max Schwarzer, Ankesh Anand, Rishab Goel, R Devon Hjelm, Aaron Courville, and Philip Bachman. Data-efficient reinforcement learning with self-predictive representations. *arXiv preprint arXiv:2007.05929*, 2020.

[35] Rémi Munos, Tom Stepleton, Anna Harutyunyan, and Marc Bellemare. Safe and efficient off-policy reinforcement learning. *Advances in neural information processing systems*, 29, 2016.

[36] John Schulman, Philipp Moritz, Sergey Levine, Michael Jordan, and Pieter Abbeel. High-dimensional continuous control using generalized advantage estimation. *arXiv preprint arXiv:1506.02438*, 2015.

[37] Guy E Blelloch. Prefix sums and their applications. 1990.

[38] Noam Shazeer. Glu variants improve transformer. *arXiv preprint arXiv:2002.05202*, 2020.

[39] Yoshua Bengio, Nicholas Léonard, and Aaron Courville. Estimating or propagating gradients through stochastic neurons for conditional computation. *arXiv preprint arXiv:1308.3432*, 2013.

[40] Zhen Qin, Xiaodong Han, Weixuan Sun, Dongxu Li, Lingpeng Kong, Nick Barnes, and Yiran Zhong. The devil in linear transformer. *arXiv preprint arXiv:2210.10340*, 2022.

[41] Biao Zhang and Rico Sennrich. Root mean square layer normalization. *Advances in Neural Information Processing Systems*, 32, 2019.

[42] Junyoung Chung, Caglar Gulcehre, KyungHyun Cho, and Yoshua Bengio. Empirical evaluation of gated recurrent neural networks on sequence modeling. *arXiv preprint arXiv:1412.3555*, 2014.

[43] Irina Higgins, Loic Matthey, Arka Pal, Christopher Burgess, Xavier Glorot, Matthew Botvinick, Shakir Mohamed, and Alexander Lerchner. beta-vae: Learning basic visual concepts with a constrained variational framework. In *International conference on learning representations*, 2016.

[44] Richard S Sutton, David McAllester, Satinder Singh, and Yishay Mansour. Policy gradient methods for reinforcement learning with function approximation. *Advances in neural information processing systems*, 12, 1999.

[45] Peter M Kogge and Harold S Stone. A parallel algorithm for the efficient solution of a general class of recurrence equations. *IEEE transactions on computers*, 100(8):786–793, 1973.

[46] Fei Deng, Junyeong Park, and Sungjin Ahn. Facing off world model backbones: Rnns, transformers, and s4. *arXiv preprint arXiv:2307.02064*, 2023.

[47] Mohammad Reza Samsami, Artem Zholus, Janarthanan Rajendran, and Sarath Chandar. Mastering memory tasks with world models. In *Second Agent Learning in Open-Endedness Workshop*, 2023.

[48] Ziyu Wang, Tom Schaul, Matteo Hessel, Hado Hasselt, Marc Lanctot, and Nando Freitas. Dueling network architectures for deep reinforcement learning. In *International conference on machine learning*, pages 1995–2003. PMLR, 2016.

[49] Thomas Hubert, Julian Schrittwieser, Ioannis Antonoglou, Mohammadamin Barekatain, Simon Schmitt, and David Silver. Learning and planning in complex action spaces. In *International Conference on Machine Learning*, pages 4476–4486. PMLR, 2021.

[50] Tuomas Haarnoja, Aurick Zhou, Pieter Abbeel, and Sergey Levine. Soft actor-critic: Off-policy maximum entropy deep reinforcement learning with a stochastic actor. In *International conference on machine learning*, pages 1861–1870. PMLR, 2018.

[51] Denis Yarats, Rob Fergus, Alessandro Lazaric, and Lerrel Pinto. Mastering visual continuous control: Improved data-augmented reinforcement learning. *arXiv preprint arXiv:2107.09645*, 2021.

[52] Julian Schrittwieser, Ioannis Antonoglou, Thomas Hubert, Karen Simonyan, Laurent Sifre, Simon Schmitt, Arthur Guez, Edward Lockhart, Demis Hassabis, Thore Graepel, et al. Mastering atari, go, chess and shogi by planning with a learned model. *Nature*, 588(7839):604–609, 2020.

[53] Samuel R Bowman, Luke Vilnis, Oriol Vinyals, Andrew M Dai, Rafal Jozefowicz, and Samy Bengio. Generating sentences from a continuous space. *arXiv preprint arXiv:1511.06349*, 2015.

[54] Carlos E Garcia, David M Prett, and Manfred Morari. Model predictive control: Theory and practice—a survey. *Automatica*, 25(3):335–348, 1989.

[55] Yuping Luo, Huazhe Xu, Yuanzhi Li, Yuandong Tian, Trevor Darrell, and Tengyu Ma. Algorithmic framework for model-based deep reinforcement learning with theoretical guarantees. *arXiv preprint arXiv:1807.03858*, 2018.

[56] Jian Shen, Han Zhao, Weinan Zhang, and Yong Yu. Model-based policy optimization with unsupervised model adaptation. *Advances in Neural Information Processing Systems*, 33:2823–2834, 2020.

[57] Marc Deisenroth and Carl E Rasmussen. Pilco: A model-based and data-efficient approach to policy search. In *Proceedings of the 28th International Conference on machine learning (ICML-11)*, pages 465–472, 2011.

[58] Jacob Buckman, Danijar Hafner, George Tucker, Eugene Brevdo, and Honglak Lee. Sample-efficient reinforcement learning with stochastic ensemble value expansion. *Advances in neural information processing systems*, 31, 2018.

[59] Vladimir Feinberg, Alvin Wan, Ion Stoica, Michael I Jordan, Joseph E Gonzalez, and Sergey Levine. Model-based value estimation for efficient model-free reinforcement learning. *arXiv preprint arXiv:1803.00101*, 2018.

[60] Albert Gu, Karan Goel, and Christopher Ré. Efficiently modeling long sequences with structured state spaces. *arXiv preprint arXiv:2111.00396*, 2021.

[61] Yi Tay, Mostafa Dehghani, Samira Abnar, Yikang Shen, Dara Bahri, Philip Pham, Jinfeng Rao, Liu Yang, Sebastian Ruder, and Donald Metzler. Long range arena: A benchmark for efficient transformers. *arXiv preprint arXiv:2011.04006*, 2020.

[62] Rudolph Emil Kalman. A new approach to linear filtering and prediction problems. 1960.

[63] Ramin Hasani, Mathias Lechner, Tsun-Hsuan Wang, Makram Chahine, Alexander Amini, and Daniela Rus. Liquid structural state-space models. *arXiv preprint arXiv:2209.12951*, 2022.

[64] Imanol Schlag, Kazuki Irie, and Jürgen Schmidhuber. Linear transformers are secretly fast weight programmers. In *International Conference on Machine Learning*, pages 9355–9366. PMLR, 2021.

[65] Jürgen Schmidhuber. Learning to control fast-weight memories: An alternative to dynamic recurrent networks. *Neural Computation*, 4(1):131–139, 1992.

[66] Hao Peng, Nikolaos Pappas, Dani Yogatama, Roy Schwartz, Noah A Smith, and Lingpeng Kong. Random feature attention. *arXiv preprint arXiv:2103.02143*, 2021.

[67] Huanru Henry Mao. Fine-tuning pre-trained transformers into decaying fast weights. *arXiv preprint arXiv:2210.04243*, 2022.

[68] Zhen Qin, Songlin Yang, and Yiran Zhong. Hierarchically gated recurrent neural network for sequence modeling. *Advances in Neural Information Processing Systems*, 36, 2024.

[69] Blazej Osinski, Chelsea Finn, Dumitru Erhan, George Tucker, Henryk Michalewski, Konrad Czechowski, Lukasz Mieczyslaw Kaiser, Mohammad Babaeizadeh, Piotr Kozakowski, Piotr Milos, et al. Model-based reinforcement learning for atari. *ICLR*, 1:2, 2020.

# A  Related Work

Model-based reinforcement learning (MBRL) methods aim to construct a world model of the real environment and utilize the model to enhance the performance of policy. The crucial aspect of MBRL lies in its utilization of a world model. Previous methods in MBRL have employed world models in various ways, including searching for optimal action sequences [54, 4, 3], generating synthetic data [1, 55, 3, 5, 56], or improving the value estimation [57, 4, 58, 59]. Our PaMoRL framework builds upon DreamerV3 [17], which embraces the Dyna paradigm [1, 2, 14]. The world model is utilized to interact with the policy to generate synthetic data, aiding model-free RL methods in maximizing cumulative task rewards during the policy learning stage.

To maximize sample efficiency, MBRL researchers have attempted to employ Recurrent Neural Networks (RNNs) [15, 16, 3, 6, 17] or Transformers [19, 21, 20, 22] as the architecture of world models. However, the update of the hidden state of the RNNs involves full matrix multiplication and the presence of nonlinearities within the recurrence hinders parallel computation, resulting in slower training speed. While Transformers provide a viable high-performance option, there is a quadratic relationship between their computational complexity and sequence length, which introduces additional computational and memory overhead.

Recent research has introduced a novel linear RNN architecture with simplified interaction between hidden states called the Structured State Space sequence model (S4) [60] that surpass both Transformers and RNNs in Long-Range Arena benchmarks (LRA) [61]. The S4 model and its variants are designed to effectively handle tasks involving long-range reasoning and draw inspiration from classical continuous-time linear state space models (SSMs) [62], which are well-established components of control theory. The relationship between the time and the frequency domain implies that SSMs have a convolutional view when the decay rate is data-independent, and therefore, training can be accomplished using the fast Fourier transforms (FFT).

We remark on the importance of incorporating a data-dependent decay rate, which is ignored by current works in SSMs until liquid S4 [63] and Mamba [33]. Our PWM builds upon linear attention with a data-dependent decay rate, which does not have the convolutional view and thus cannot use FFT for training but allows the use of parallel scans. The field of linear attentions and linear RNNs exhibits a close relationship [31], i.e. linear attentions can be reformulated as linear RNNs during auto-regressive decoding, revealing similarities to the update rules observed in fast weight additive outer products [64, 65]. These updated rules can be seen as a special case of element-wise linear recurrence. However, this formulation in linear attention cannot forget irrelevant information, resulting in the attention dilution issue. To address this limitation, gating mechanisms [66, 67, 68] can be used to facilitate the forgetting of irrelevant information similar to those in traditional RNNs.

The work that is most similar to our PaMoRL is Mamba [33]. Both our PaMoRL and Mamba have data-dependent decay rates and employ parallel scans to speed up the training process. However, there are significant differences between PaMoRL and Mamba in terms of model architectures and hardware preferences. In terms of model architecture, Mamba needs to maintain self-consistency with previous work in the SSM family, and therefore it must adhere to the paradigm of classical state space models, representing continuous differential equations. It needs to be parameterized and discretized using special tricks to achieve the gating mechanism implicitly. On the other hand, we recognize this limitation of Mamba and use a more "simple yet effective" gating mechanism. Regarding hardware preference, Mamba employs a special IO-aware parallel scanning algorithm for efficient training, which focuses on reducing the number of reads and writes between SRAM and HBM in the GPU through kernel fusion, and is suitable for improving the training efficiency of the hardware features when the model architecture is determined. In contrast, to satisfy the need for flexibility in MBRL, the parallel scanner we use is inspired by high-performance computing hardware design and focuses more on generality. Our parallel scanning method is compatible with arbitrary model architectures, as long as it satisfies the parallelization conditions mentioned in Section 2, as compared to the model architecture-specific parallel scanning method used by Mamba.

# B  Illustrations to Parallel Scan Algorithms

## B.1  Kogge-stone Scanner

A common example of such a first-order recurrence problem is a time-varying linear system, the system's state at timestep $t$ is $x_t$, computed from the system's internal dynamical variables $a_t$ and $b_t$, as shown in Equation 9. Depending on the problem, the variables $a_t$ and $b_t$ can be real or complex numbers, constants, etc.

$$
\begin{aligned}
x_1 &= b_1, \\
x_2 &= a_2 x_1 + b_2, \\
x_3 &= a_3 x_2 + b_3, \\
&\vdots \\
x_i &= a_i x_{i-1} + b_i, \\
&\vdots \\
x_L &= a_L x_{L-1} + b_L.
\end{aligned}
\tag{9}
$$

Before solving the problem, we can define the function $A(m, n)$ and $B(m, n)$, as shown in Equation 12.

$$
\begin{aligned}
A(m, n) &= \prod_{j=n}^{m} a_i, \\
B(m, n) &= \sum_{i=n}^{m} (\prod_{j=i+1}^{m} a_j) b_i, \text{ where } n \le m.
\end{aligned}
\tag{10}
$$

Now we can integrate Equation 9 with Equation 10 to get Equation 11.

$$
\begin{aligned}
B(1,1) &= x_1 = b_1, \\
B(2,1) &= x_2 = a_2 x_1 + b_2 = a_2 B(1,1) + B(2,2) = A(2,2)B(1,1) + B(2,2), \\
&\vdots \\
B(4,1) &= x_4 = a_4 x_3 + b_4 = a_4 a_3 B(2,1) + B(4,3) = A(4,3)B(2,1) + B(4,3), \\
&\vdots \\
B(2i,1) &= x_{2i} = (\prod_{j=i+1}^{2i} a_j) B(i,1) + B(2i, i+1) = A(2i, i+1)B(i,1) + B(2i, i+1).
\end{aligned}
\tag{11}
$$

It can be observed in Equation 11 that $B(2i, i+1)$ is associated with the computation of $A(2i, i+1)$ but independent from $B(i, 1)$, which means that we can split the computation of $B(2i, 1)$ into two parallel parts. For reasons of notational simplicity, we define the tuple $Q(m, n)$ that wraps the functions $A(m, n)$ and $B(m, n)$, as shown in Equation 12.

$$
Q(m, n) = (A(m, n), B(m, n)), \text{ where } n \le m.
\tag{12}
$$

Figure 6 shows the operation of the Kogge-stone scanner [45] when the sequence length $L = 8$. After $\lceil \log_2 L \rceil$ iterations the solution to the problem $x_1, \ldots, x_T$ can be computed.

## B.2  Odd-even Scanner

To avoid the extra computational complexity of $\log_2 L$ generated by the Kogge-stone scanner [45], the Odd-even scanner [28] uses an algorithmic pattern that arises often in parallel computing: balanced

trees. The idea is to build a balanced binary tree on the input data and start scanning from the root. A binary tree with $L$ leaves has $\log_2 L$ layers with $2^d$ nodes per layer $d \in [0, L)$. If we perform one operation on each node, then we will perform $\mathcal{O}(L)$ operations in one traversal of the tree. The tree we construct is not an actual data structure, but rather a concept that we use to determine what each thread has to do at each step of the traversal.

The algorithm consists of two phases: up-sweep and down-sweep. During the up-sweep phase, we traverse from the leaves to the root of the tree. During the down-sweep phase, we backtrack from the root node up the tree, using the results computed in the up-sweep phase. Figure 7 shows the operation of the Kogge-stone scanner [28] when the sequence length $L = 8$.

Note that since this is an exclusive scan (i.e., the sum is not included in the result), we zero out the last element of the array between phases. This zero is propagated back to the head of the array in the down-sweep phase. This scanning algorithm performs $\mathcal{O}(2L)$ operations, so it is very efficient.

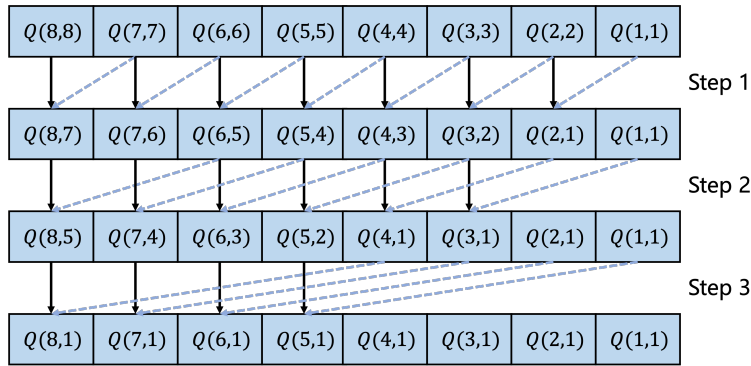

Figure 6: Illustrations of the operation of the Kogge-stone scanner when the sequence length $L = 8$.

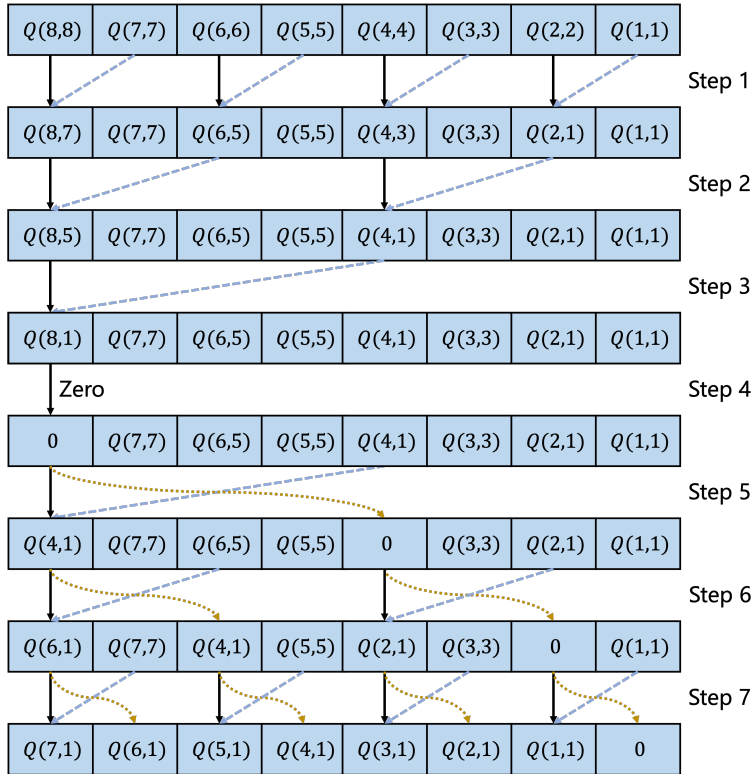

Figure 7: Illustrations of the operation of the Odd-even scanner when the sequence length $L = 8$.

# C  Training Curves of the Atari 100K Benchmark

The Atari 100K benchmark [69] is a standard RL benchmark comprising 26 Atari games featuring diverse gameplay mechanics. It is designed to assess a broad spectrum of agent skills, and agents are limited to executing 400 thousand discrete actions within each environment, which is approximately equivalent to 2 hours of human gameplay. To put this in perspective, when there are no constraints on sample efficiency, the typical practice is to train agents for 200M steps.

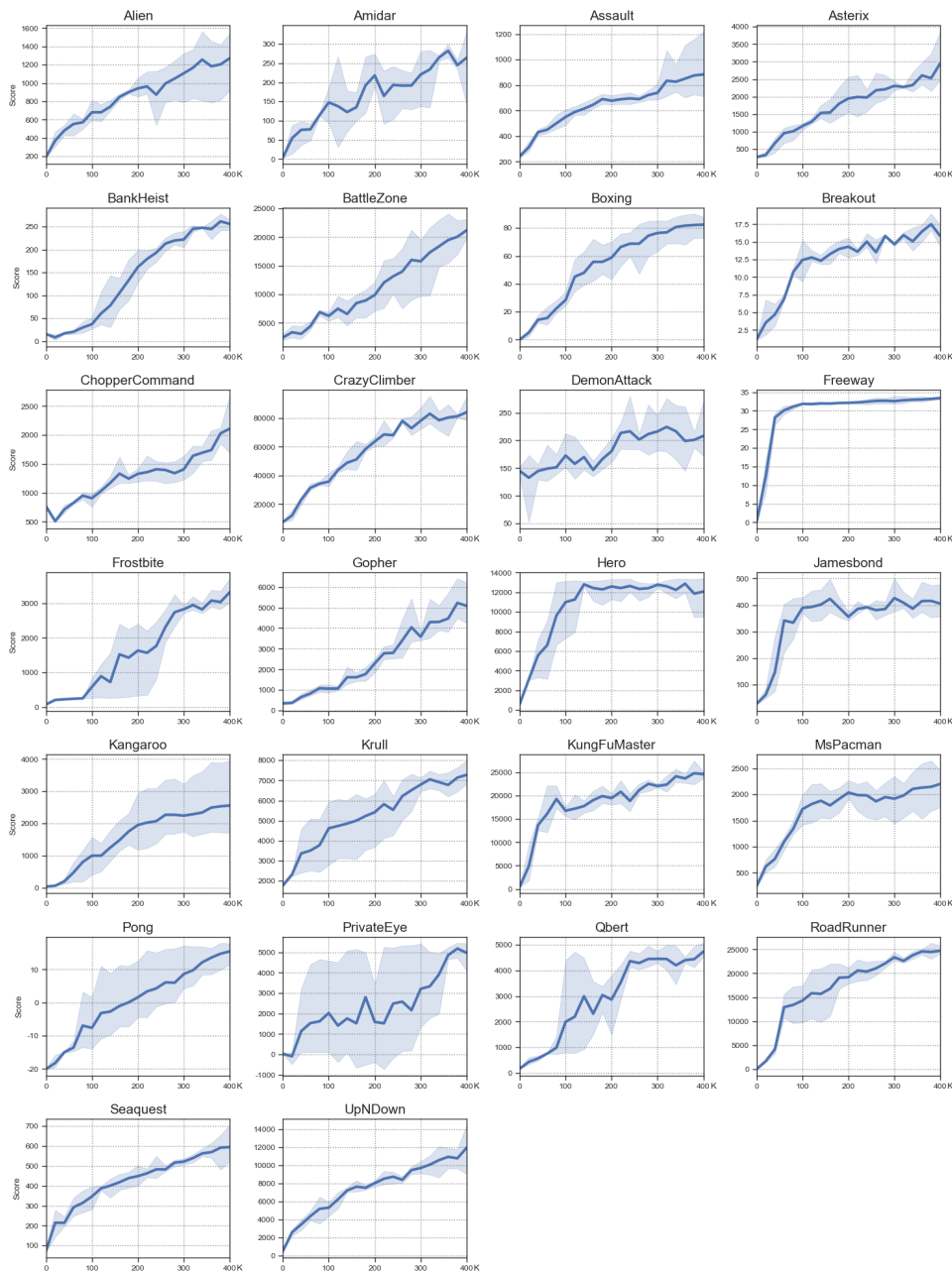

Figure 8: Training curves on the Atari 100k benchmark. The solid line represents the average result over 5 seeds, and the filled area indicates the range between the maximum and minimum results across these 5 seeds.

# D  Training Curves of the DeepMind Control Suite

DeepMind Control suite [24] is a standard RL benchmark comprising various tasks with continuous action spaces. It supports both image observation and low-dimensional proprioception observation. When there are no constraints on sample efficiency, the typical practice is to train agents for millions of steps.

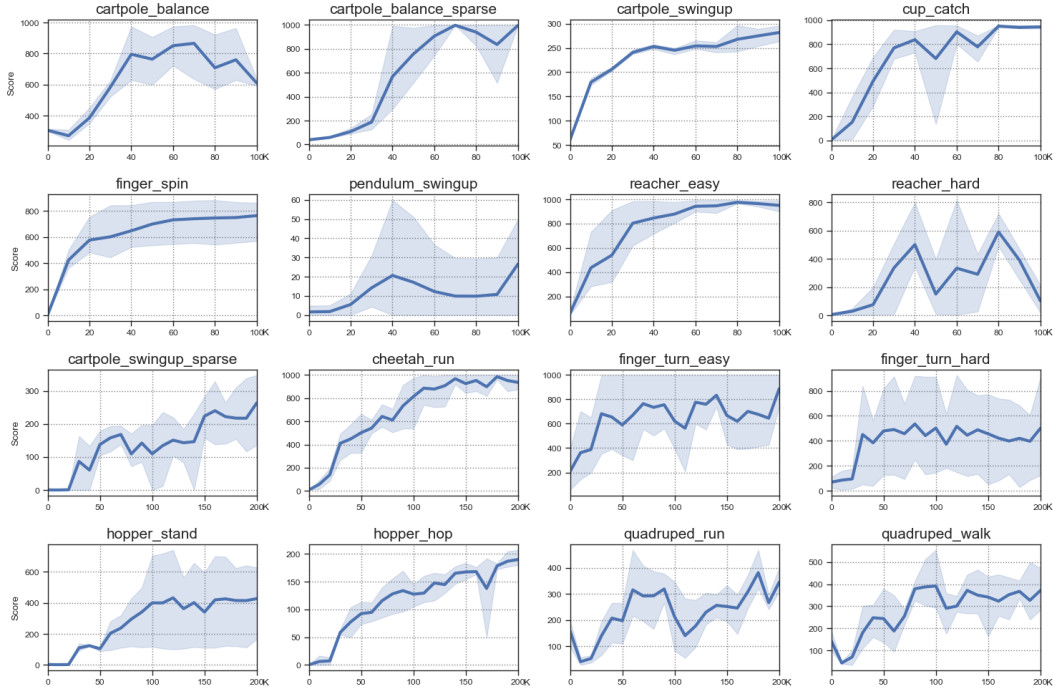

Figure 9: Training curves on the DeepMind Control suite with image observations. The solid line represents the average result over 5 seeds, and the filled area indicates the range between the maximum and minimum results across these 5 seeds.

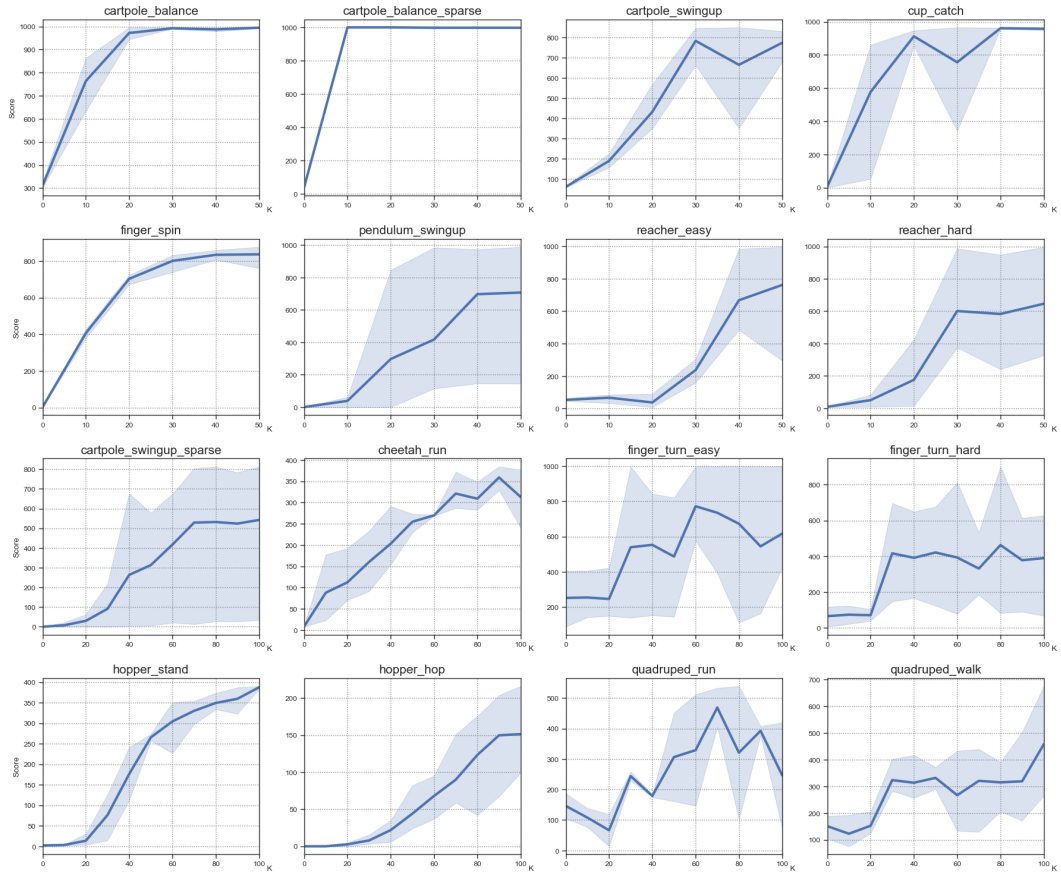

Figure 10: Training curves on the DeepMind Control suite with proprioception observations. The solid line represents the average result over 5 seeds, and the filled area indicates the range between the maximum and minimum results across these 5 seeds.

# E  Details of Model Architecture

Table 4: Architecture details of the image encoder. The size of the modules is omitted and can be derived from the shape of the tensors. SiLU refers to the sigmoid-weighted linear units used for activation, while Linear represents a fully connected layer. Flatten and Reshape operations are employed to alter the tensor's indexing method while preserving the data and their original order. Conv denotes a CNN layer characterized by kernel $= 4$, stride $= 2$, and padding $= 1$. BN denotes the batch normalization layer.

| Module | Output tensor shape |
|---|---|
| Imput image ($o_t$) | $3 \times 64 \times 64$ |
| Conv1+BN1+SiLU | $32 \times 32 \times 32$ |
| Conv2+BN2+SiLU | $64 \times 16 \times 16$ |
| Conv3+BN3+SiLU | $128 \times 8 \times 8$ |
| Conv4+BN4+SiLU | $256 \times 4 \times 4$ |
| Flatten | 4096 |
| Linear + BN5 + Reshape | $32 \times 32$ |

Table 5: Architecture details of the image decoder. DeConv denotes a transpose CNN layer characterized by kernel $= 4$, stride $= 2$, and padding $= 1$.

| Module | Output tensor shape |
|---|---|
| Random samples ($z_t$) | $32 \times 32$ |
| Flatten | 1024 |
| Linear+BN1+SiLU | 4096 |
| Reshape | $256 \times 4 \times 4$ |
| DeConv1+BN1+SiLU | $128 \times 8 \times 8$ |
| DeConv2+BN2+SiLU | $64 \times 16 \times 16$ |
| DeConv3+BN3+SiLU | $32 \times 32 \times 32$ |
| DeConv4 | $3 \times 64 \times 64$ |

Table 6: Action mixer. Concatenate denotes combining the last dimension of two tensors and merging them into one new tensor. The variable $A$ represents the action dimension. $D$ denotes the feature dimension of the sequence model. LN is an abbreviation for layer normalization.

| Module | Output tensor shape |
|---|---|
| Random samples ($z_t$) & Action ($a_t$) | $32 \times 32, A$ |
| Reshape and concatenate | $1024 + A$ |
| Linear+LN1+SiLU | $D$ |
| Linear+LN2 | $D$ |

Table 7: Modules which are pure MLPs. 1-layer MLP corresponds to a fully connected layer. 255 is the size of the bucket of symlog two-hot loss [17]. $K$ refers to the dimension of proprioception observations.

| Name | MLP layers | Iutput/Hidden/Output shape |
|---|---|---|
| Encoder (proprio) | 3 | $K$/512/$D$ |
| Decoder (proprio) | 3 | $D$/512/$K$ |
| Dynamic predictor | 1 | $D$/$D$/1024 |
| Reward predictor | 3 | $D$/$D$/225 |
| Continuation predictor | 3 | $D$/$D$/1 |
| Actor network | 3 | $D$/$D$/$A$ |
| Critic network | 3 | $D$/$D$/225 |

# F  Hyperparameters

Table 8: Full hyperparameters. Note that the environment will provide a "done" signal when losing a life but will continue running until the actual reset occurs. This life information configuration aligns with the setup used in IRIS [20]. Regarding data sampling, each time, we sample $B_1$ trajectories of length $T$ for world model training and sample $B_2$ trajectories of length $C$ for starting the imaginations.

| Hyperparameter | Symbol | Value |
|---|---|---|
| Sequence model layers | $K$ | 2 |
| Hidden size of query, key, and value | - | 64 |
| Hidden size of sequence model output | $D$ | 512 |
| World model training batch size | $B_1$ | 16 |
| World model training batch length | $T$ | 64 |
| Imagination batch size | $B_2$ | 1024 |
| Imagination horizon | $H$ | 16 |
| Update world model every environment step | - | 1 |
| Update policy environment env step | - | 1 |
| Scan algorithm for world model training | - | Odd-even |
| Scan algorithm for policy training | - | Kogge-stone |
| Gamma | $\gamma$ | $0.997^{\text{action repeat}}$ |
| Lambda | $\lambda$ | 0.95 |
| Entropy coefficiency | $\eta$ | $3 \times 10^{-4}$ |
| Optimizer | - | Adam |
| World model learning rate | - | $1 \times 10^{-4}$ |
| World model gradient norm clipping | - | 100.0 |
| Actor-critic learning rate | - | $3 \times 10^{-5}$ |
| Actor-critic gradient norm | - | 100.0 |
| Gray image input | - | False |
| Frame stacking | - | False |
| Frame skipping | - | 4 (Atari) or 2 (DMControl) |
| Use of life information | - | True (Atari) |

# G Pytorch-style Pseudo-code of Parallel Scan

## G.1 Odd-even scanner

```python
def odd_even_parallel_scan(inputs, operator):
    """
    Odd/Even Parallel Scanner.
    Inputs:
        inputs: tuple of sequence elements.
        operator: binary operator function.
    Outputs:
        outputs: tuple of sequence elements.
    """
    Length = inputs[0].shape[0]

    if Length < 2:
        return inputs

    reduced_inputs = operator(
        (input[:-1][0::2] for input in inputs),
        (input[1::2] for input in inputs)
    )
    odd_inputs = odd_even_parallel_scan(reduced_inputs, operator)

    if Length % 2 == 0:
        even_inputs = operator(
            (input[:-1] for input in odd_inputs),
            (input[2::2] for input in inputs)
            )
    else:
        even_inputs = operator(
            (input for input in odd_inputs),
            (input[2::2] for input in inputs)
        )

    even_inputs = [
        torch.cat((input[0:1], even_input), dim=0)
        for (input, even_input) in zip(inputs, even_inputs)
    ]

    outputs = [
        interleave(odd_input, even_input)
        for (even_input, odd_input) in zip(even_inputs, odd_inputs)
    ]
    return outputs

def interleave(odd, even):
    padded_odd = torch.cat((odd, torch.zeros_like(odd[-1:])), dim=0)
    outputs = torch.stack((even, padded_odd[:even.shape[0]]), dim=1)
    outputs = outputs.flatten(0, 1)[:(odd.shape[0] + even.shape[0])]
    return outputs
```

### G.2 Kogge-stone scanner

```python
def kogge_stone_parallel_scan(inputs, operator):
    """
    Kogge-Stone Parallel Scanner.
    Inputs:
        inputs: tuple of sequence elements.
        operator: binary operator function.
    Outputs:
        outputs: tuple of sequence elements.
    """
    Length = inputs[0].shape[0]
    Times = math.ceil(math.log2(Length))

    for i in range(Times):
        interval = int(2 ** i)
        outputs = operator(
            (input[:-interval] for input in inputs),
            (input[interval:] for input in inputs)
        )
        inputs = [
            torch.cat((input[:interval], output), dim=0)
            for (input, output) in zip(inputs, outputs)
        ]
    return inputs
```

## H Pytorch-style Pseudo-code of Parallelized Eligibility Trace Estimation

```python
def parallel_eligibility_trace(reward, value, next_value, p_cont, lam):
    """
    Parallel Eligibility Trace Estimations.
    """
    ones = torch.ones_like(reward)
    p_cont, lam = p_cont * ones, lam * ones
    lam = torch.cat((lam[1:], ones[:1]), dim=0)

    delta = reward + p_cont * next_value - value
    flipped_delta = delta.flip(dims=(0,))
    flipped_lam = (p_cont * lam).flip(dims=(0,))

    residual = odd_even_parallel_scan(
        [flipped_lam, flipped_delta], binary_return_fn)
    returns = value + residual[1].flip(dims=(0,))
    return returns

def parallel_lambda_return(reward, value, next_value, p_cont, lam):
    """
    Parallel TD-Lambda Estimations.
    """
    ones = torch.ones_like(reward)
    p_cont, lam = p_cont * ones, lam * ones

    delta = reward + p_cont * next_value * (1 - lam)
    last = delta[-1:] + p_cont[-1:] * lam[-1:] * next_value[-1:]
    delta = torch.cat((delta[:-1], last), dim=0)

    flipped_delta = delta.flip(dims=(0,))
    flipped_lam = (p_cont * lam).flip(dims=(0,))

    returns = odd_even_parallel_scan(
        [flipped_lam, flipped_delta], binary_return_fn)
    returns = returns[1].flip(dims=(0,))
    return returns

def binary_return_fn(cur_i, cur_j):
    coef_i, in_i = cur_i
    coef_j, in_j = cur_j
    return coef_i * coef_j, coef_j * in_i + in_j
```

# I Additional Comparisons on the Atari 100K Benchmark

Atari 100K benchmark [69] is a standard RL benchmark comprising 26 Atari games featuring diverse gameplay mechanics. It is designed to assess a broad spectrum of agent skills, and agents are limited to executing 400 thousand discrete actions within each environment, which is approximately equivalent to 2 hours of human gameplay. To put this in perspective, when there are no constraints on sample efficiency, the typical practice is to train agents for 200M steps.

In this section, we compare the performance of our PaMoRL framework with planning-based methods such as EfficientZero [7] and EfficientZero V2 [9] and methods with much larger networks, i.e., BBF [8] on the Atari 100K benchmark. The full results are shown in Table I. The PaMoRL framework is not as good as the other methods in terms of the number of superhuman games, median score, and average score. However, it leads the pack of 13/26 games in terms of an individual game perspective.

Table 9: Experimental results on the 26 games of Atari 100k after 2 hours of real-time experience and human-normalized aggregate metrics. Bold and underlined numbers indicate the highest and the second-highest scores, respectively.

| Game | Random | Human | EfficientZero | BBF | EfficientZero V2 | PaMoRL (Ours) |
|---|---|---|---|---|---|---|
| Alien | 227.8 | 7127.7 | 808.5 | 1173.2 | **1557.7** | 1270.6 |
| Amidar | 5.8 | 1719.5 | 148.6 | 244.6 | 184.9 | **264.4** |
| Assault | 222.4 | 742 | 1263.1 | **2098.5** | 1757.5 | 833.8 |
| Asterix | 210 | 8503.3 | 25557.8 | 3946.1 | **61810** | 2957.3 |
| BankHeist | 14.2 | 753.1 | 351 | 732.9 | **1316.7** | 225.9 |
| BattleZone | 2360 | 37187.5 | 13871.2 | **24459.8** | 14433.3 | 23120 |
| Boxing | 0.1 | 12.1 | 52.7 | 85.8 | 75 | **87.9** |
| Breakout | 1.7 | 30.5 | **414.1** | 370.6 | 400.1 | 15.8 |
| ChopperCommand | 811 | 7387.8 | 1117.3 | **7549.3** | 1196.6 | 2110.7 |
| CrazyClimber | 10780.5 | 35829.4 | 83940.2 | 58431.8 | **112363.3** | 84102 |
| DemonAttack | 152.1 | 1971 | 13003.9 | 13341.4 | **22773.5** | 208.2 |
| Freeway | 0 | 29.6 | 21.8 | 25.5 | 0 | **33.8** |
| Frostbite | 65.2 | 4334.7 | 296.3 | 2384.8 | 1136.3 | **3711.4** |
| Gopher | 257.6 | 2412.5 | 3260.3 | 1331.2 | 3868.7 | **5085.2** |
| Hero | 1027 | 30826.4 | 9315.9 | 7818.6 | 9705 | **12076.2** |
| Jamebond | 29 | 302.8 | 517 | **1129.6** | 468.3 | 405 |
| Kangaroo | 52 | 3035 | 724.1 | **6614.7** | 1886.7 | 2554.7 |
| Krull | 1598 | 2665.5 | 5663.3 | 8223.4 | **9080** | 7273.2 |
| KungFuMaster | 258.5 | 22736.3 | **30944.8** | 18991.7 | 28883.3 | 24624.7 |
| MsPacman | 307.3 | 6951.6 | 1281.2 | 2008.3 | **2251** | 2201.7 |
| Pong | -20.7 | 14.6 | 20.1 | 16.7 | **20.8** | 15.5 |
| PrivateEye | 24.9 | 69571.3 | 96.7 | 40.5 | 99.8 | **4968.6** |
| Qbert | 163.9 | 13455 | 14448.5 | 4447.1 | **16058.3** | 4730.3 |
| Roadrunner | 11.5 | 7845 | 17751.3 | **33426.8** | 27516.7 | 24726.7 |
| Seaquest | 68.4 | 42054.7 | 1100.2 | 1232.5 | **1974** | 595.2 |
| UpNDwon | 533.4 | 11693.2 | **17264.2** | 12101.7 | 15224.3 | 11953.8 |
| Games >Human | 0 | 26 | 14 | 12 | **21** | 9 |
| Median | 0% | 100% | 111.53% | 91.71% | **123.47%** | 71.75% |
| Mean | 0% | 100% | 194.46% | 224.74% | **267.97%** | 126.64% |

## J Runtime of Experiments

Table 10: Average runtime of experiments

| Task | Atari 100K | Proprio (easy) | Proprio (hard) | Vision (easy) | Vision (hard) |
|---|---|---|---|---|---|
| Runtime | 3.5 hours | 0.94 hours | 1.88 hours | 2.74 hours | 7.1 hours |

## K Effectiveness of Batch Normalization Trick

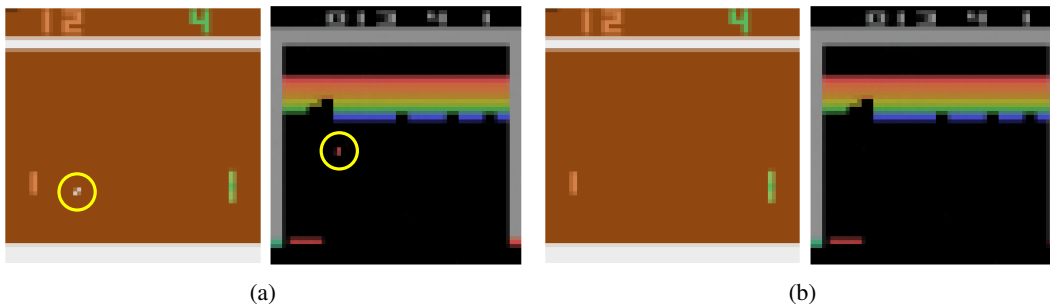

(a)                                  (b)

Figure 11: Visualizations on Batch Normalization trick in *Pong* and *Breakout*.

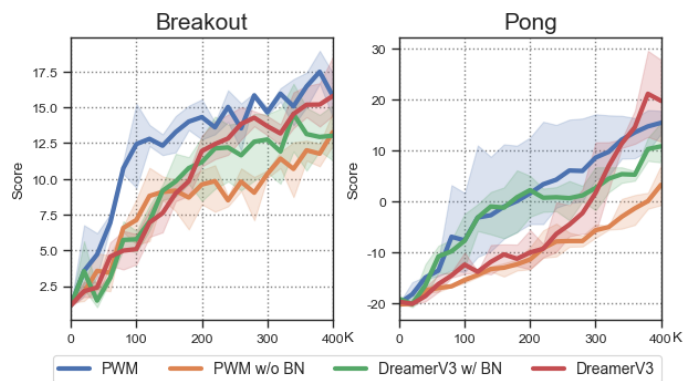

Figure 12: Quantitative results on the effectiveness of the Batch Normalization trick.

## L  Video Predictions

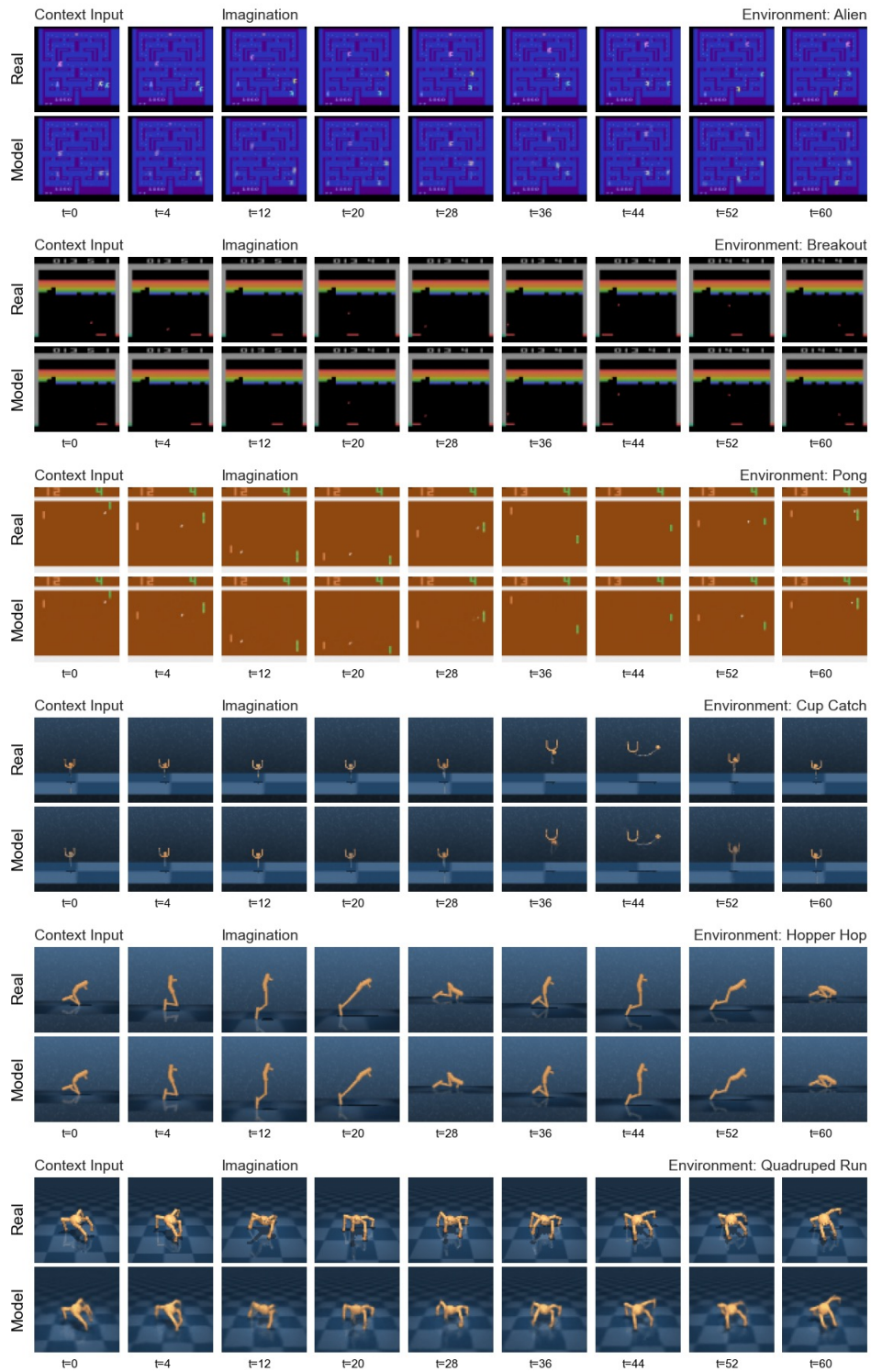

Figure 13:  Multi-step predictions on several environments in Atari games and DeepMind Control suite. The world model utilizes 5 observations and actions as contextual input, enabling the imagination of future events spanning 56 frames in an auto-regressive manner.

# M Initializations in *Freeway*

The reward function in *Freeway* is sparse since the agent is only rewarded when it completely crosses the road. In addition, bumping into cars will drag it down, preventing it from smoothly ascending the highway. This poses an exploration problem for newly initialized agents because a random policy will almost surely never obtain a non-zero reward with a 100k frames budget. The solution to this problem is actually straightforward and requires stretches of time when the "UP" action is oversampled. In this paper, we opted for the simplest strategy of having an initialized buffer with fulfilled "UP" actions. Hence, we don't need to lowered the sampling temperature to avoid random walks that would not be conducive to learning in the early stages of training. Consequently, once it received its first few rewards through exploration, our PaMoRL could internalize the sparse reward function in its world model.

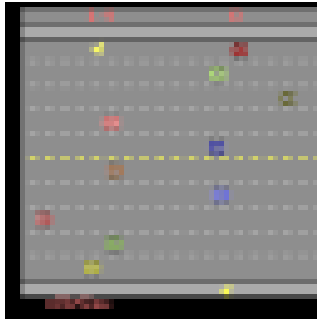

Figure 14: A game of *Freeway*. Cars will bump the player down, making it very unlikely to cross the road and be rewarded for random policies.

